# STIMULUS ENCODING BY MULTIDIMENSIONAL RECEPTIVE FIELDS IN SINGLE CELLS AND CELL POPULATIONS IN V1 OF AWAKE MONKEY

**Edward Stern**
Center for Neural Computation
and Department of Neurobiology
Life Sciences Institute
Hebrew University
Jerusalem, Israel

**Ad Aertsen**
Institut fur Neuroinformatik
Ruhr-Universitat-Bochum
Bochum, Germany

**Eilon Vaadia**
Center for Neural Computation
and Physiology Department
Hadassah Medical School
Hebrew University
Jerusalem, Israel

**Shaul Hochstein**
Center for Neural Computation
and Department of Neurobiology,
Life Sciences Institute
Hebrew University
Jerusalem, Israel

## ABSTRACT

Multiple single neuron responses were recorded from a single electrode in V1 of alert, behaving monkeys. Drifting sinusoidal gratings were presented in the cells' overlapping receptive fields, and the stimulus was varied along several visual dimensions. The degree of dimensional separability was calculated for a large population of neurons, and found to be a continuum. Several cells showed different temporal response dependencies to variation of different stimulus dimensions, i.e. the tuning of the modulated firing was not necessarily the same as that of the mean firing rate. We describe a multidimensional receptive field, and use simultaneously recorded responses to compute a multi-neuron receptive field, describing the information processing capabilities of a group of cells. Using dynamic correlation analysis, we propose several computational schemes for multidimensional spatiotemporal tuning for groups of cells. The implications for neuronal coding of stimuli are discussed.

## INTRODUCTION

The receptive field is perhaps the most useful concept for understanding neuronal information processing. The ideal definition of the receptive field is that set of stimuli which cause a change in the neuron's firing properties. However, as with many such concepts, the use of the receptive field in describing the behavior of sensory neurons falls short of the ideal. The classical method for describing the receptive field has been to measure the "tuning curve" i.e. the response of the neuron as a function of the value of one dimension of the stimulus. This presents a problem because the sensory world is multidimensional: For example, even a simple visual stimulus, such as a patch of a sinusoidal grating, may vary in location, orientation, spatial frequency, temporal frequency, movement direction and speed, phase, contrast, color, etc. Does the tuning to one dimension remain constant when other dimensions are varied? i.e. are the dimensions linearly separable? It is not unreasonable to expect inseparability: Consider an oriented, spatially discrete receptive field. The excitation generated by passing a bar through the receptive field will of course change with orientation. However, the shape of this tuning curve will depend upon the bar width, related to the spatial frequency. This effect has not been studied quantitatively, however. If interactions among dimensions exist, do they account for a large portion of the cell's response variance? Are there discrete populations of cells, with some cells showing interactions among dimensions and others not? These question have clear implications for the problem of neural coding.

Related to the question of dimensional separability is that of stimulus encoding: Given that the receptive field is multidimensional in nature, how can the cell maximize the amount of stimulus information it encodes? Does the neuron use a single code to represent all the stimulus dimensions? It is possible that interactions lead to greater uncertainty in stimulus identification. Does the small number of visual cortical cells encode all the possible combinations of stimuli using only spike rate as the dependent variable? We present data indicating that more information is indeed present in the neuronal response, and propose a new approach for its utilization.

The final problem that we address is the following: Clearly, many cells participate in the stimulus encoding process. Arriving at a valid concept of a multidimensional receptive field, can we generalize this concept to more than one cell introducing the notion of a multi-cellular receptive field?

## METHODS

Drifting sinusoidal gratings were presented for 500 msec to the central 10 degrees of the visual field of monkeys performing a fixation task. The gratings were varied in orientation, spatial frequency, temporal frequency, and movement direction. We recorded from up to 3 cells simultaneously with a single electrode in the monkey's primary visual cortex (V1). The cells described in this study were well separated, using a template-matching procedure. The responses of the neurons were plotted as Peri-Stimulus Time Histograms (PSTHs) and their parameters quantified (Abeles, 1982), and offline Fourier analysis and time-dependent crosscorrelation analysis (Aertsen et al, 1989) were performed.

## RESULTS

Recording the responses of visual cortical neurons to stimuli varied over a number of dimensions, we found that in some cases, the tuning curve to one dimension depended on the value of another dimension. Figure 1A shows the spatial-frequency tuning curve of a single cell measured at 2 different stimulus orientations. When the orientation of the stimulus is 72 degrees, the peak response is at a spatial frequency of 4.5 cycles/degree (cpd), while at an orientation of 216 degrees, the spatial frequency of peak response is 2.3 cpd. If the responses to different visual dimensions were truly linearly separable, the tuning curve to any single dimension would have the same shape and, in particular, position of peak, despite any variations in other dimensions. If the tuning curves are not parallel, then interactions must exist between dimensions. Clearly, this is an example of a cell whose responses are not linearly separable. In order to quantify the inseparability phenomenon, analyses of variance were performed, using spike rate as the dependent variable, and the visual dimensions of the stimuli as the independent variables. We then measured the amount of interaction as a percentage of the total between-conditions

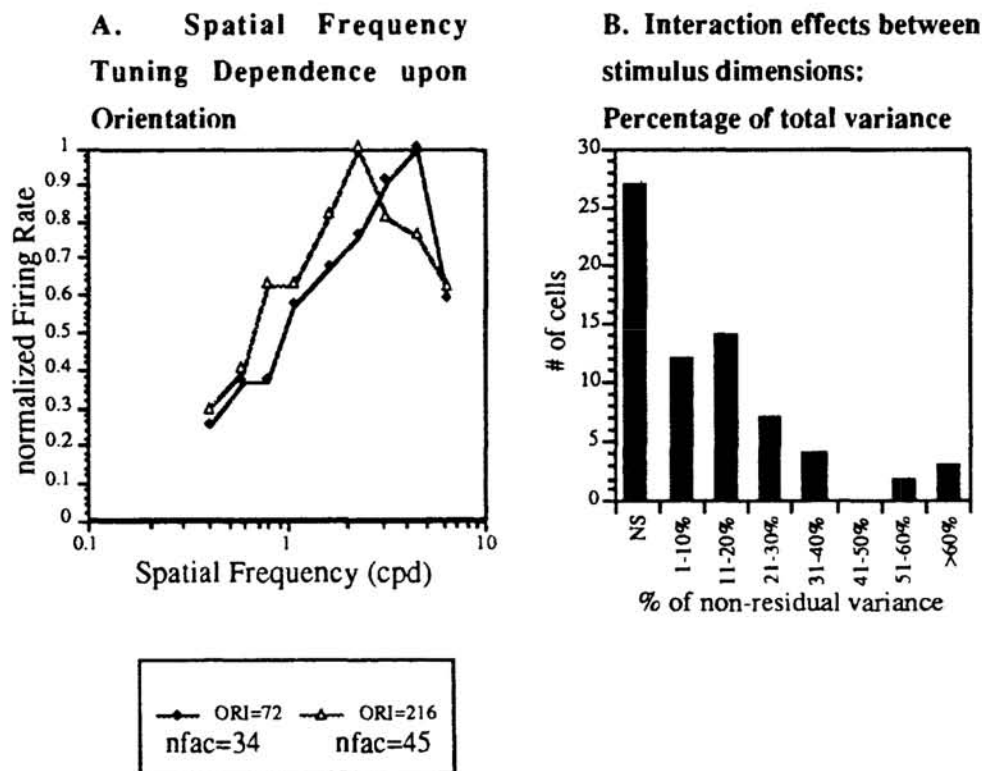

Figure 1: Dimensional Inseparability of Visual Cortical Neurons. A: An example of dimensionsional inseparability in the response of a single cell; B: Histogram of dimensional inseparability as a percentage of total response variance.

variance divided by the residuals. The resulting histogram for 69 cells is shown in Figure 1B. Although there are several cells with non-significant interactions, i.e. linearly separable dimensions, this is not the majority of cells. The amount of dimensional inseparability seems to be a continuum. We suggest that separability is a significant variable in the coding capability of the neurons, which must be taken into account when modeling the representation of sensory information by cortical neural networks.

We found that the time course of the response was not always constant, but varied with stimulus parameters. Cortical cell responses may have components which are sustained (constant over time), transient (with a peak near stimulus onset and/or offset), or modulated (varying with the stimulus period). For example, Figure 2 shows the responses of a single neuron in V1 to 50 stimuli, varying in orientation and spatial frequency. Each response is plotted as a PSTH, and the stippled bar under the PSTH indicates the time of

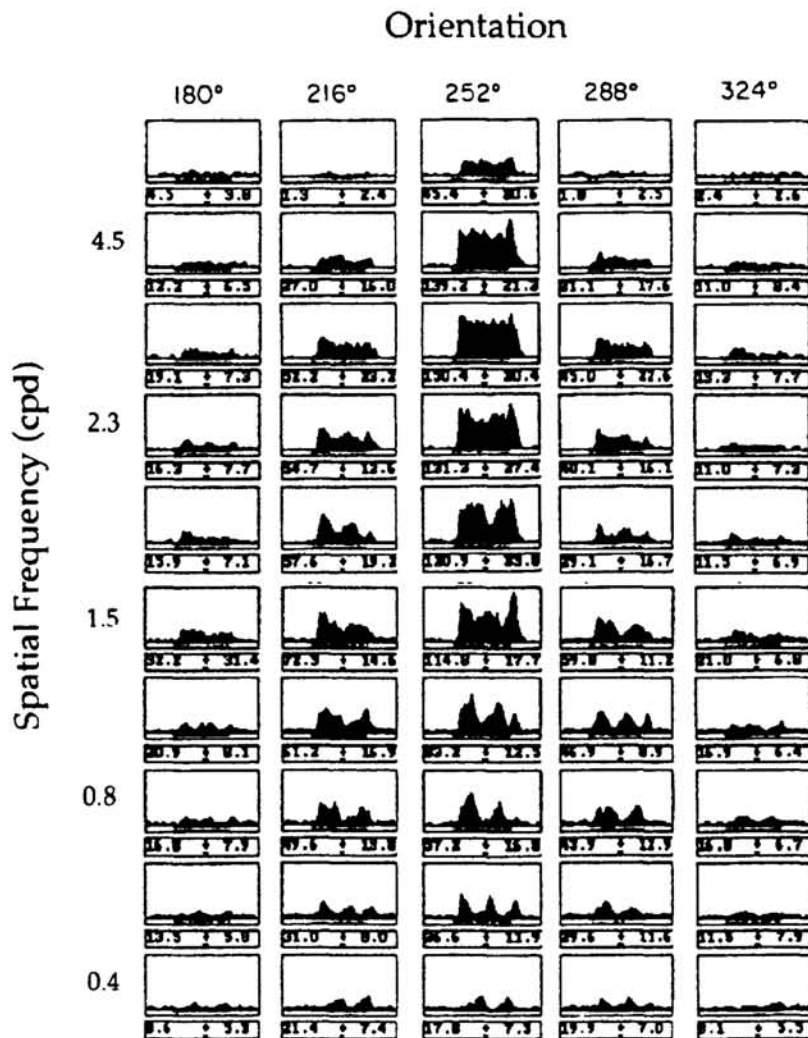

**Figure 2: Spatial Frequency/Orientation Tuning of Responses of V1 Cell**

the stimulus presentation (500 msec). The numbers beneath each PSTH are the firing rate averaged over the response time, and the standard deviations of the response over repetitions of the stimulus (in this case 40). Clearly, the cell is orientation selective, and the neuronal response is also tuned to spatial frequency. The stimulus eliciting the highest firing rate is ORI=252 degrees; SF=3.2 cycles/degree (cpd). However, when looking at the responses to lower spatial frequencies, we see a modulation in the PSTH. The modulation, when present, has 2 peaks, corresponding to the temporal frequency of the stimulus grating (4 cycles/second). Therefore, although the response rate of the cell is lower at low spatial frequencies than for other stimuli, the spike train carries additional information about another stimulus dimension.

If the visual neuron is considered as a linear system, the predicted response to a drifting sinusoidal grating would be a (rectified) sinusoid of the same (temporal) frequency as that of the stimulus, i.e. a modulated response (Enroth-Cugell & Robson, 1966; Hochstein & Shapley, 1976; Spitzer & Hochstein, 1988). However, as seen in Figure 2, in some stimulus regimes the cell's response deviates from linearity. We conclude that the linearity or nonlinearity of the response is dependent upon the stimulus conditions (Spitzer & Hochstein, 1985). A modulated response is one that would be expected from simple cells, while the sustained response seen at higher spatial frequencies is that expected from complex cells. Our data therefore suggest that the simple/complex cell categorization is not complete.

A further example of response time-course dependence on stimulus parameters is seen in Figure 3A. In this case, the stimulus was varied in spatial frequency and temporal frequency, while other dimensions were held constant. Again, as spatial frequency is raised, the modulation of the PSTH gives way to a more sustained response. Furthermore, as temporal frequency is raised, both the sustained and the modulated responses are replaced by a single transient response. When present, the frequency of the modulation follows that of the temporal frequency of the stimulus. Fourier analysis of the response histograms (Figure 3B) reveals that the DC and fundamental component (FC) are not tuned to the same stimulus values (arrows indicating peaks). We propose that this information may be available to the cell readout, enabling the single cell to encode multiple stimulus dimensions simultaneously.

Thus, a complete description of the receptive field must be multidimensional in nature. Furthermore, in light of the evidence that the spike train is not constant, one of the dimensions which must be used to display the receptive field must be time.

Figure 4 shows one method of displaying a multidimensional response map, with time along the abscissa (in 10 msec bins) and orientation along the ordinate. In the top two figures, the z axis, represented in gray-scale, is the number of counts (spikes) per bin. Therefore, each line is a PSTH, with counts (bin height) coded by shading. In this example, cell 2 (upper picture) is tuned to orientation, with peaks at 90 and 270 degrees. The cell is only slightly direction selective, as represented by the fact that the 2 areas of high activity are similarly shaded. However, there is a transient peak at 270 degrees which

**A.**

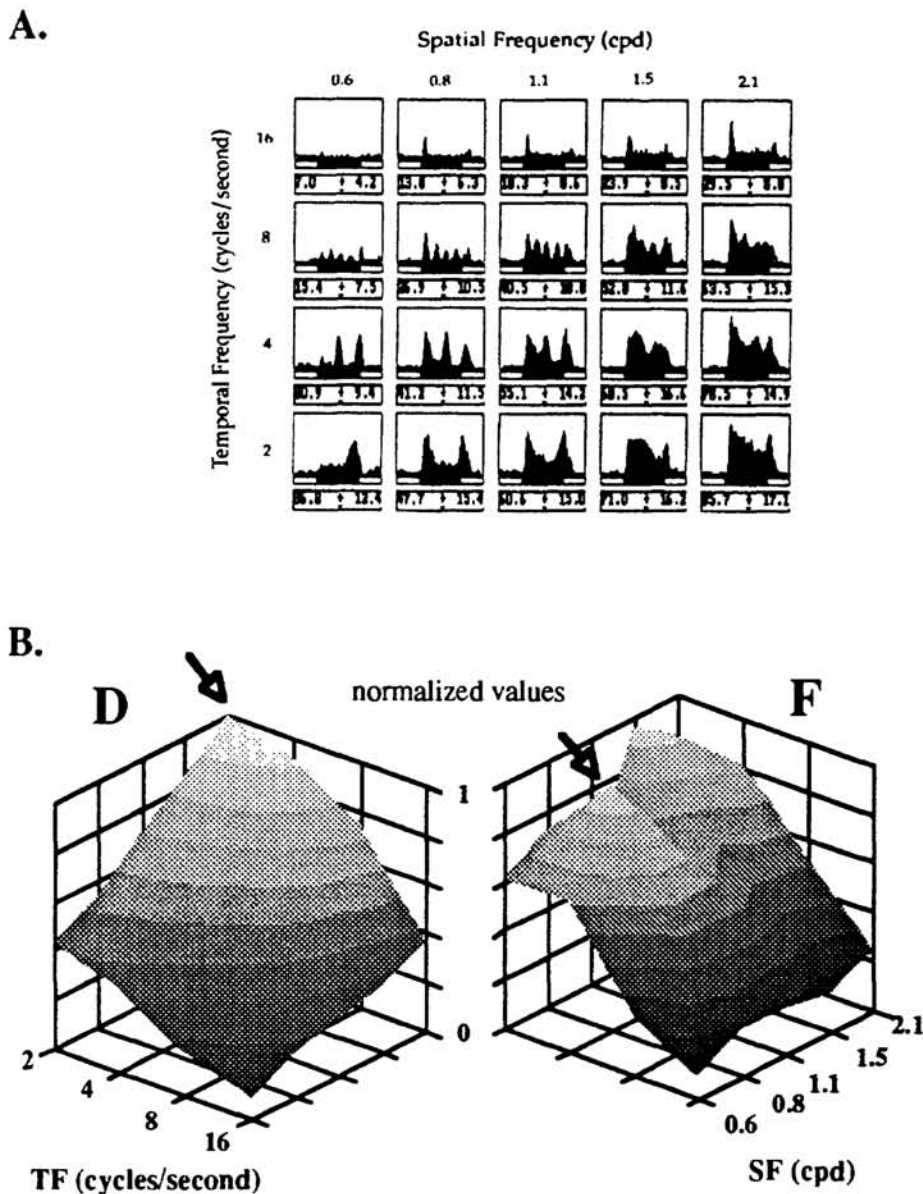

**Figure 3: A. TF/SF Tuning of response of V1 cell.
B. Tuning of DC and FC of response to stimulus parameters.**

is absent at 90 degrees. The middle picture, representing a simultaneously recorded cell shows a different pattern of activity. The orientation tuning of this cell is similar to that of cell 2, but it has stronger directional selectivity, (towards 90 degrees). In this case, the transient is also at 90 degrees. The bottom picture shows the joint activity of these 2 cells. Rather than each line being a PSTH, each line is a Joint PSTH (JPSTH; Aertsen et al, 1989). This histogram represents the time-dependent correlated activity of a pair of cells. It is equivalent to sliding a window across a spike train of one neuron and

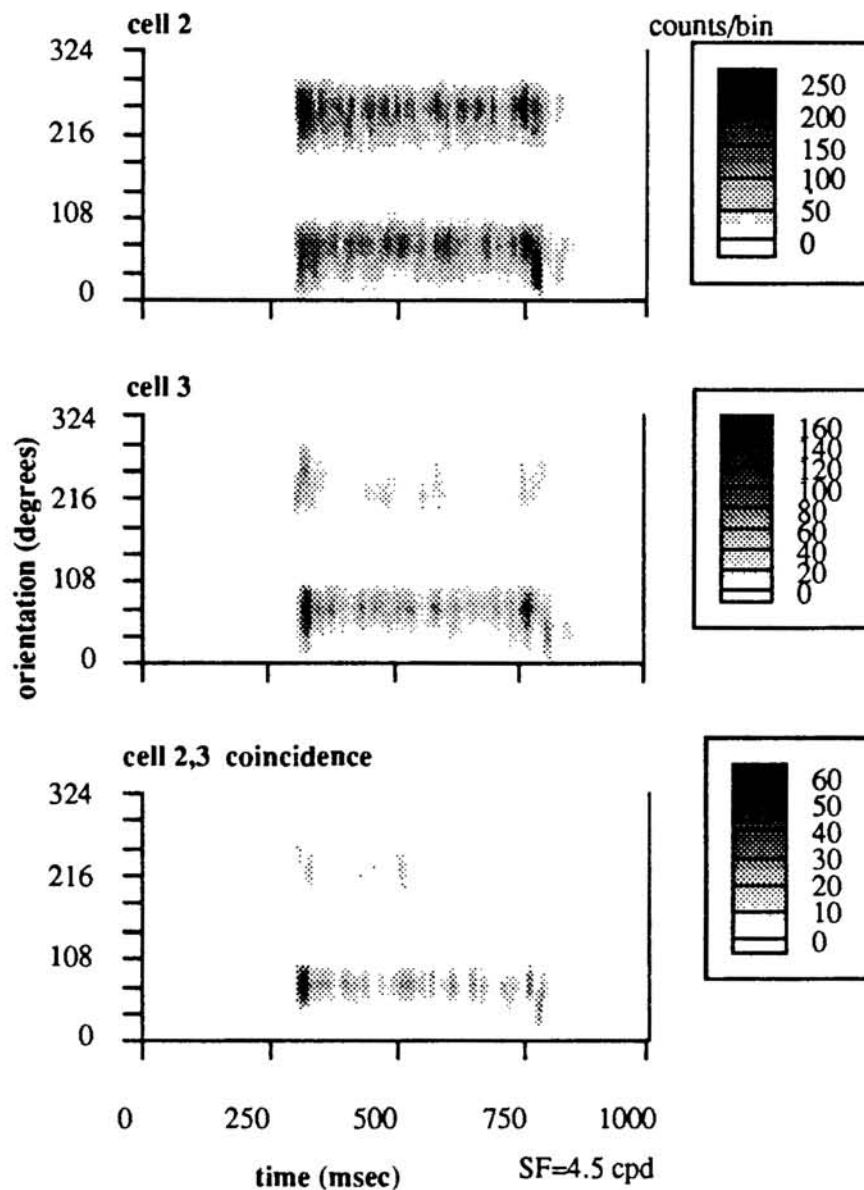

**Figure 4: Response Maps.**

**Top, Middle: Single-cell Multidimensional Receptive Fields;**

**Bottom: Multi-Cell Multidimensional Receptive Field**

asking when a spike from another neuron falls within the window. The size of the window can be varied; here we used 2 msec. Therefore, we are asking when these cells fire within 2 msec of each other, and how this is connected to the stimulus. The z axis is now coincidences per bin. We may consider this the logical AND activity of these cells; if there is a cell receiving information from both of these neurons, this is the receptive field which would describe its input. Clearly, it is different from the each of the 2 individual cells. In our results, it is more narrowly tuned, and the tuning can not be predicted from the individual components. We emphasize that this is the "raw" JPSTH, which is not corrected for stimulus effects, common input, or normalized. This is because we want a measure comparable to the PSTHs themselves, to compare a multi-unit

receptive field to its single unit components. In this case, however, a significant (p<0.01; Palm et al, 1988) "mono-directional" interaction is present. For a more complete description of the receptive field, this type of figure, shown here for one spatial frequency only, can be shown for all spatial frequencies as "slices" along a fourth axis. However, space limitations prevent us from presenting this multidimensional aspect of the multicellular receptive field.

## CONCLUSIONS

We have shown that interactions among stimulus dimensions account for a significant proportion of the response variance of V1 cells. The variance of the interactions itself may be a useful parameter when considering a population response, as the amount and location of the dimensional inseparability varies among cells. We have also shown that different temporal characteristics of the spike trains can be tuned to different dimensions, and add to the encoding capabilities of the cell in a neurobiologically realistic manner. Finally, we use these results to generate multidimensional receptive fields, for single cells and small groups of cells. We emphasize that this can be generalized to larger populations of cells, and to compute the population responses of cells that may be meaningful for the cortex as a biological neuronal network.

### Acknowledgements

We thank Israel Nelken, Hagai Bergman, Volodya Yakovlev, Moshe Abeles, Peter Hillman, Robert Shapley and Valentino Braitenberg for helpful discussions. This study was supported by grants from the U.S.-Israel Bi-National Science Foundation (BSF) and the Israel Academy of Sciences.

### References

1. Abeles, M. Quantification, Smoothing, and Confidence Limits for Single Units' Histograms *J. Neurosci. Methods 5* , 317-325, 1982.

2. Aertsen, A.M.H.J., Gerstein, G. L., Habib, M.K., and Palm, G. Dynamics of Neuronal Firing Correlation: Modulation of "Effective Connectivity" *J. Neurophysiol 51* (5), 900-917, 1989.

3. Enroth-Cugell, C. and Robson, J.G. The Contrast Sensitivity of Retinal Ganglion Cells of the Cat *J Physiol. Lond 187*, 517-552, 1966.

4. Hochstein, S. and Shapley, R. M. Linear and Nonlinear Spatial Subunits in Y Cat Retinal Ganglion Cells *J Physiol. Lond 262*, 265-284, 1976.

5. Palm, G., Aertsen, A.M.H.J. and Gerstein, G.L. On the Significance of Correlations Among Neuronal Spike Trains *Biol. Cybern. 59* , 1-11, 1988.

6. Spitzer, H. and Hochstein, S. Simple and Complex-Cell Response Dependencies on Stimulation Parameters *J.Neurophysiol 53*, 1244-1265, 1985.

7. Spitzer, H. and Hochstein, S. Complex Cell Receptive Field Models *Prog. in Neurobiology. 31* , 285-309, 1988.